# A Non-Parametric Approach to Dynamic Programming

**Oliver B. Kroemer**[1,2]     **Jan Peters**[1,2]

[1]Intelligent Autonomous Systems, Technische Universität Darmstadt
[2]Robot Learning Lab, Max Planck Institute for Intelligent Systems
{kroemer,peters}@ias.tu-darmstadt.de

## Abstract

In this paper, we consider the problem of policy evaluation for continuous-state systems. We present a non-parametric approach to policy evaluation, which uses kernel density estimation to represent the system. The true form of the value function for this model can be determined, and can be computed using Galerkin's method. Furthermore, we also present a unified view of several well-known policy evaluation methods. In particular, we show that the same Galerkin method can be used to derive Least-Squares Temporal Difference learning, Kernelized Temporal Difference learning, and a discrete-state Dynamic Programming solution, as well as our proposed method. In a numerical evaluation of these algorithms, the proposed approach performed better than the other methods.

## 1   Introduction

Value functions are an essential concept for determining optimal policies in both optimal control [1] and reinforcement learning [2, 3]. Given the value function of a policy, an improved policy is straightforward to compute. The improved policy can subsequently be evaluated to obtain a new value function. This loop of computing value functions and determining better policies is known as *policy iteration*. However, the main bottleneck in policy iteration is the computation of the value function for a given policy. Using the Bellman equation, only two classes of systems have been solved exactly: tabular discrete state and action problems [4] as well as linear-quadratic regulation problems [5]. The exact computation of the value function remains an open problem for most systems with continuous state spaces [6]. This paper focuses on steps toward solving this problem.

As an alternative to exact solutions, approximate policy evaluation methods have been developed in reinforcement learning. These approaches include *Monte Carlo* methods, *temporal difference* learning, and *residual gradient* methods. However, Monte Carlo methods are well-known to have an excessively high variance [7, 2], and tend to overfit the value function to the sampled data [2]. When using function approximations, temporal difference learning can result in a biased solution[8]. Residual gradient approaches are biased unless multiple samples are taken from the same states [9], which is often not possible for real continuous systems.

In this paper, we propose a non-parametric method for continuous-state policy evaluation. The proposed method uses a kernel density estimate to represent the system in a flexible manner. Model-based approaches are known to be more data efficient than direct methods, and lead to better policies [10, 11]. We subsequently show that the true value function for this model has a *Nadaraya-Watson kernel regression* form [12, 13]. Using Galerkin's projection method, we compute a closed-form solution for this regression problem. The

resulting method is called *Non-Parametric Dynamic Programming* (NPDP), and is a stable as well as consistent approach to policy evaluation.

The second contribution of this paper is to provide a unified view of several sample-based algorithms for policy evaluation, including the NPDP algorithm. In Section 3, we show how *Least-Squares Temporal Difference learning* (LSTD) in [14], *Kernelized Temporal Difference learning* (KTD) in [15], and *Discrete-State Dynamic Programming* (DSDP) in [4, 16] can all be derived using the same Galerkin projection method used to derive NPDP. In Section 4, we compare these methods using empirical evaluations.

In reinforcement learning, the uncontrolled system is usually represented by a Markov Decision Process (MDP). An MDP is defined by the following components: a set of states $\mathbb{S}$; a set of actions $\mathbb{A}$; a transition distribution $p(\boldsymbol{s}'|\boldsymbol{a},\boldsymbol{s})$, where $\boldsymbol{s}' \in \mathbb{S}$ is the next state given action $\boldsymbol{a} \in \mathbb{A}$ in state $\boldsymbol{s} \in \mathbb{S}$; a reward function $r$, such that $r(\boldsymbol{s},\boldsymbol{a})$ is the immediate reward obtained for performing action $\boldsymbol{a}$ in state $\boldsymbol{s}$; and a discount factor $\gamma \in [0,1)$ on future rewards. Actions $\boldsymbol{a}$ are selected according to the stochastic policy $\pi(\boldsymbol{a}|\boldsymbol{s})$. The goal is to maximize the discounted rewards that are obtained; i.e., $\max \sum_{t=0}^{\infty} \gamma^t \boldsymbol{r}(\boldsymbol{s}_t, \boldsymbol{a}_t)$. The term *system* will refer jointly to the agent's policy and the MDP.

The value of a state $V(\boldsymbol{s})$, for a specific policy $\pi$, is defined as the expected discounted sum of rewards that an agent will receive after visiting state $\boldsymbol{s}$ and executing policy $\pi$; i.e.,

$$V(\boldsymbol{s}) = E\left\{ \sum_{t=0}^{\infty} \gamma^t \boldsymbol{r}(\boldsymbol{s}_t, \boldsymbol{a}_t) \big| \, \boldsymbol{s}_0 = \boldsymbol{s}, \pi \right\}. \tag{1}$$

By using the Markov property, Eq. (1) can be rewritten as the *Bellman equation*

$$V(\boldsymbol{s}) = \int_{\mathbb{A}} \int_{\mathbb{S}} \pi(\boldsymbol{a}|\boldsymbol{s}) \, p(\boldsymbol{s}'|\boldsymbol{s}, \boldsymbol{a}) \left[ r(\boldsymbol{s}, \boldsymbol{a}) + \gamma V^{\pi}(\boldsymbol{s}') \right] \mathrm{d}\boldsymbol{s}' \mathrm{d}\boldsymbol{a}. \tag{2}$$

The advantage of using the Bellman equation is that it describes the relationship between the value function at one state $\boldsymbol{s}$ and its immediate follow-up states $\boldsymbol{s}' \sim p(\boldsymbol{s}'|\boldsymbol{s}, \boldsymbol{a})$. In contrast, the direct computation of Eq. (1) relies on the rewards obtained from entire trajectories.

## 2 Non-Parametric Model-based Dynamic Programming

We begin describing the NPDP approach by introducing the kernel density estimation framework used to represent the system. The true value function for this model has a kernel regression form, which can be computed by using Galerkin's projection method. We subsequently discuss some of the properties of this algorithm, including its consistency.

### 2.1 Non-Parametric System Modeling

The dynamics of a system are compactly represented by the joint distribution $p(\boldsymbol{s}, \boldsymbol{a}, \boldsymbol{s}')$. Using Bayes rule and marginalization, one can compute the transition probabilities $p(\boldsymbol{s}'|\boldsymbol{s}, \boldsymbol{a})$ and the current policy $\pi(\boldsymbol{a}|\boldsymbol{s})$ from this joint distribution; e.g. $p(\boldsymbol{s}'|\boldsymbol{s}, \boldsymbol{a}) = p(\boldsymbol{s}, \boldsymbol{a}, \boldsymbol{s}') / \int p(\boldsymbol{s}, \boldsymbol{a}, \boldsymbol{s}') \mathrm{d}\boldsymbol{s}'$. Rather than assuming that certain prior information is given, we will focus on the problem where only sampled information of the system is available. Hence, the system's joint distribution is modeled from a set of $n$ samples obtained from the real system. The $i^{\text{th}}$ sample includes the current state $\boldsymbol{s}_i \in \mathbb{S}$, the selected action $\boldsymbol{a}_i \in \mathbb{A}$, and the follow-up state $\boldsymbol{s}_i' \in \mathbb{S}$, as well as the immediate reward $r_i \in \mathbb{R}$. The state space $\mathbb{S}$ and the action space $\mathbb{A}$ are assumed to be continuous.

We propose using *kernel density estimation* to represent the joint distribution [17, 18] in a non-parametric manner. Unlike parametric models, non-parametric approaches use the collected data as features, which leads to accurate representations of arbitrary functions [19]. The system's joint distribution is therefore modeled as $p(\boldsymbol{s}, \boldsymbol{a}, \boldsymbol{s}') = n^{-1} \sum_{i=1}^{n} \psi_i(\boldsymbol{s}') \varphi_i(\boldsymbol{a}) \phi_i(\boldsymbol{s})$, where $\psi_i(\boldsymbol{s}') = \psi(\boldsymbol{s}', \boldsymbol{s}_i')$, $\varphi_i(\boldsymbol{a}) = \varphi(\boldsymbol{a}, \boldsymbol{a}_i)$, and $\phi_i(\boldsymbol{s}) = \phi(\boldsymbol{s}, \boldsymbol{s}_i)$ are symmetric kernel functions. In practice, the kernel functions $\psi$ and $\phi$ will often be the same. To ensure a valid probability density, each kernel must integrate to one; i.e., $\int \phi_i(\boldsymbol{s}) \, \mathrm{d}s = 1$, $\forall i$, and similarly for $\psi$ and $\varphi$. As an additional constraint, the kernel must always be positive; i.e., $\psi_i(\boldsymbol{s}') \varphi_i(\boldsymbol{a}) \phi_i(\boldsymbol{s}) \geq 0$, $\forall \boldsymbol{s} \in \mathbb{S}$. This representation implies a factorization into separate $\psi_i(\boldsymbol{s}')$, $\varphi_i(\boldsymbol{a})$, and $\phi_i(\boldsymbol{s})$ kernels. As a result, an individual sample cannot express correlations between $\boldsymbol{s}'$, $\boldsymbol{a}$, and $\boldsymbol{s}$. However, the representation does allow multiple samples to express correlations between these components in $p(\boldsymbol{s}, \boldsymbol{a}, \boldsymbol{s}')$.

The reward function $r(\boldsymbol{s}, \boldsymbol{a})$ must also be represented. Given the kernel density estimate representation, the expected reward for a state-action pair is denoted as [12]

$$r(\boldsymbol{s}, \boldsymbol{a}) = E[r|\boldsymbol{s}, \boldsymbol{a}] = \frac{\sum_{k=1}^{n} r_k \varphi_k(\boldsymbol{a}) \phi_k(\boldsymbol{s})}{\sum_{i=1}^{n} \varphi_i(\boldsymbol{a}) \phi_i(\boldsymbol{s})}.$$

Having specified the model of the system dynamics and rewards, the next step is to derive the corresponding value function.

## 2.2 Resulting Solution

In this section, we propose an approach to computing the value function for the continuous model specified in Section 2.1. Every policy has a unique value function, which fulfills the Bellman equation, Eq. (2), for all states [2, 20]. Hence, the goal is to solve the Bellman equation for the entire state space, and not just at the sampled states. This goal can be achieved by using the Galerkin projection method to compute the value function for the model [21].

The Galerkin method involves first projecting the integral equation into the space spanned by a set of basis functions. The integral equation is then solved in this projected space. To begin, the Bellman equation, Eq. (2), is rearranged as

$$V(\boldsymbol{s}) = \int_{\mathbb{A}} \int_{\mathbb{S}} \pi(\boldsymbol{a}|\boldsymbol{s}) r(\boldsymbol{s}, \boldsymbol{a}) p(\boldsymbol{s}'|\boldsymbol{s}, \boldsymbol{a}) \, \mathrm{d}\boldsymbol{s}' \mathrm{d}\boldsymbol{a} + \int_{\mathbb{S}} \int_{\mathbb{A}} p(\boldsymbol{s}'|\boldsymbol{s}, \boldsymbol{a}) \gamma V(\boldsymbol{s}') \pi(\boldsymbol{a}|\boldsymbol{s}) \, \mathrm{d}\boldsymbol{a} \mathrm{d}\boldsymbol{s}',$$

$$p(\boldsymbol{s}) V(\boldsymbol{s}) = \int_{\mathbb{A}} p(\boldsymbol{a}, \boldsymbol{s}) r(\boldsymbol{s}, \boldsymbol{a}) \, \mathrm{d}\boldsymbol{a} + \gamma \int_{\mathbb{S}} p(\boldsymbol{s}', \boldsymbol{s}) V(\boldsymbol{s}') \, \mathrm{d}\boldsymbol{s}'. \qquad (3)$$

Before applying the Galerkin method, we derive the exact form of the value function. Expanding the reward function and joint distributions, as defined in Section 2.1, gives

$$p(\boldsymbol{s}) V(\boldsymbol{s}) = n^{-1} \int_{\mathbb{A}} \sum_{k=1}^{n} \varphi_k(\boldsymbol{a}) \phi_k(\boldsymbol{s}) \frac{\sum_{i=1}^{n} r_i \varphi_i(\boldsymbol{a}) \phi_i(\boldsymbol{s})}{\sum_{j=1}^{n} \varphi_j(\boldsymbol{a}) \phi_j(\boldsymbol{s})} \, \mathrm{d}\boldsymbol{a} + \gamma \int_{\mathbb{S}} p(\boldsymbol{s}', \boldsymbol{s}) V(\boldsymbol{s}') \, \mathrm{d}\boldsymbol{s}',$$

$$p(\boldsymbol{s}) V(\boldsymbol{s}) = \int_{\mathbb{A}} n^{-1} \sum_{i=1}^{n} r_i \varphi_i(\boldsymbol{a}) \phi_i(\boldsymbol{s}) \, \mathrm{d}\boldsymbol{a} + \gamma \int_{\mathbb{S}} n^{-1} \sum_{i=1}^{n} \psi_i(\boldsymbol{s}') \phi_i(\boldsymbol{s}) V(\boldsymbol{s}') \, \mathrm{d}\boldsymbol{s}',$$

$$p(\boldsymbol{s}) V(\boldsymbol{s}) = n^{-1} \sum_{i=1}^{n} r_i \phi_i(\boldsymbol{s}) + n^{-1} \sum_{i=1}^{n} \gamma \int_{\mathbb{S}} \psi_i(\boldsymbol{s}') \phi_i(\boldsymbol{s}) V(\boldsymbol{s}') \, \mathrm{d}\boldsymbol{s}',$$

Therefore, $p(\boldsymbol{s}) V(\boldsymbol{s}) = n^{-1} \sum_{i=1}^{n} \theta_i \phi_i(\boldsymbol{s})$, where $\theta$ are value weights. Given that $p(\boldsymbol{s}) = n^{-1} \sum_{j=1}^{n} \phi_j(\boldsymbol{s})$, the true value function of the kernel density estimate system has a Nadaraya-Watson kernel regression [12, 13] form

$$V(\boldsymbol{s}) = \frac{\sum_{i=1}^{n} \theta_i \phi_i(\boldsymbol{s})}{\sum_{j=1}^{n} \phi_j(\boldsymbol{s})}. \qquad (4)$$

Having computed the true form of the value function, the Galerkin projection method can be used to compute the value weights $\theta$. The projection is performed by taking the expectation of the integral equation with respect to each of the $n$ basis function $\phi_i$. The resulting $n$ simultaneous equations can be written as the vector equation

$$\int_{\mathbb{S}} \boldsymbol{\phi}(\boldsymbol{s}) p(\boldsymbol{s}) V(\boldsymbol{s}) \mathrm{d}\boldsymbol{s} = \int_{\mathbb{S}} \boldsymbol{\phi}(\boldsymbol{s}) n^{-1} \boldsymbol{\phi}(\boldsymbol{s})^T \boldsymbol{r} \mathrm{d}\boldsymbol{s} + \gamma \int_{\mathbb{S}} \int_{\mathbb{S}} \boldsymbol{\phi}(\boldsymbol{s}) n^{-1} \left( \boldsymbol{\phi}(\boldsymbol{s})^T \boldsymbol{\psi}(\boldsymbol{s}') \right) V(\boldsymbol{s}') \mathrm{d}\boldsymbol{s}' \mathrm{d}\boldsymbol{s},$$

where the i$^{\text{th}}$ elements of the vectors are given by $[\boldsymbol{r}]_i = r_i$, $[\boldsymbol{\phi}(\boldsymbol{s})]_i = \phi_i(\boldsymbol{s})$, and $[\boldsymbol{\psi}(\boldsymbol{s}')]_i = \psi_i(\boldsymbol{s}')$. Expanding the value functions gives

$$\int_{\mathbb{S}} \boldsymbol{\phi}(\boldsymbol{s}) \boldsymbol{\phi}(\boldsymbol{s})^T \boldsymbol{\theta} \mathrm{d}\boldsymbol{s} = \int_{\mathbb{S}} \boldsymbol{\phi}(\boldsymbol{s}) \boldsymbol{\phi}(\boldsymbol{s})^T \boldsymbol{r} \mathrm{d}\boldsymbol{s} + \gamma \int_{\mathbb{S}} \int_{\mathbb{S}} \boldsymbol{\phi}(\boldsymbol{s}) \left( \boldsymbol{\phi}(\boldsymbol{s})^T \boldsymbol{\psi}(\boldsymbol{s}') \right) \frac{\boldsymbol{\phi}(\boldsymbol{s}')^T \boldsymbol{\theta}}{\sum_{i=1}^{n} \phi_i(\boldsymbol{s}')} \mathrm{d}\boldsymbol{s}' \mathrm{d}\boldsymbol{s},$$

$$\boldsymbol{C}\boldsymbol{\theta} = \boldsymbol{C}\boldsymbol{r} + \gamma \boldsymbol{C}\boldsymbol{\lambda}\boldsymbol{\theta},$$

where $\boldsymbol{C} = \int_{\mathbb{S}} \boldsymbol{\phi}(\boldsymbol{s}) \boldsymbol{\phi}(\boldsymbol{s})^T \mathrm{d}\boldsymbol{s}$, and $\boldsymbol{\lambda} = \int_{\mathbb{S}} (\sum_{i=1}^{n} \phi_i(\boldsymbol{s}'))^{-1} \boldsymbol{\psi}(\boldsymbol{s}') \boldsymbol{\phi}(\boldsymbol{s}')^T \mathrm{d}\boldsymbol{s}'$ is a stochastic matrix; i.e., a transition matrix. The matrix $\boldsymbol{C}$ can become singular if two basis functions

**Algorithm 1** Non-Parametric Dynamic Programming

| | |
|---|---|
| INPUT: | COMPUTATION: |
| $n$ **system samples**:<br>   state $\boldsymbol{s}_i$, next state $\boldsymbol{s}'_i$, and reward $r_i$<br>**Kernel functions**:<br>  $\phi_i\left(\boldsymbol{s}_j\right) = \phi\left(\boldsymbol{s}_i, \boldsymbol{s}_j\right)$, and $\psi_i\left(\boldsymbol{s}'_j\right) = \psi\left(\boldsymbol{s}'_i, \boldsymbol{s}'_j\right)$<br>**Discount factor**:<br>  $0 \le \gamma < 1$ | **Reward vector**:<br>  $[\boldsymbol{r}]_i = r_i$<br>**Transition matrix**:<br>  $[\boldsymbol{\lambda}]_{i,j} = \int_{\mathbb{S}} \frac{\phi_j(\boldsymbol{s}')\psi_i(\boldsymbol{s}')}{\sum_{k=1}^{n}\phi_k(\boldsymbol{s}')}\mathrm{d}\boldsymbol{s}'$<br>**Value weights**:<br>  $\boldsymbol{\theta} = (\boldsymbol{I} - \gamma\boldsymbol{\lambda})^{-1}\boldsymbol{r}$ |
| OUTPUT: | |
| **Value function**:            $V\left(\boldsymbol{s}\right) = \frac{\sum_{i=1}^{n}\theta_i\phi_i(\boldsymbol{s})}{\sum_{j=1}^{n}\phi_j(\boldsymbol{s})}$ | |

are coincident. In such cases, there exists an infinite set of solutions for $\boldsymbol{\theta}$. However, all of the solutions result in identical values. The NPDP algorithm uses the solution given by

$$\boldsymbol{\theta} = (\boldsymbol{I} - \gamma\boldsymbol{\lambda})^{-1}\boldsymbol{r}, \tag{5}$$

which always exists for any stochastic matrix $\boldsymbol{\lambda}$. Thus, the derivation has shown that the exact value function for the model in Section 2.1 has a Nadaraya-Watson kernel regression form, as shown in Eq. (4), with weights $\boldsymbol{\theta}$ given by Eq. (5). The non-parametric dynamic programming algorithm is summarized in Alg. 1. The NPDP algorithm ultimately requires only the state information $\boldsymbol{s}$ and $\boldsymbol{s}'$, and not the actions $\boldsymbol{a}$. In Section 3, we will show how this form of derivation can also be used to derive the LSTD, KTD, and DSDP algorithms.

## 2.3 Properties of the NPDP Algorithm

In this section, we discuss some of the key properties of the proposed NPDP algorithm, including precision, accuracy, and computational complexity. Precision refers to how close the predicted value function is to the true value function of the model, while accuracy refers to how close the model is to the true system.

One of the key contributions of this paper is providing the true form of the value function for policy evaluation with the non-parametric model described in Section 2.1. The parameters of this value function can be computed precisely by solving Eq. (5). Even if $\boldsymbol{\lambda}$ is evaluated numerically, a high level of precision can still be obtained.

As a non-parametric method, the accuracy of the NPDP algorithm depends on the number of samples obtained from the system. It is important that the model, and thus the value function, converges to that of the true system as the number of samples increases; i.e., that the model is statistically consistent. In fact, kernel density estimation can be proven to have almost sure convergence to the true distribution for a wide range of kernels [22].

Given that $\boldsymbol{\lambda}$ is a stochastic matrix and $0 \le \gamma < 1$, it is well-known that the inversion of $(\boldsymbol{I} - \gamma\boldsymbol{\lambda})$ is well-defined [16]. The inversion can therefore also be expanded according to the Neumann series; i.e., $\boldsymbol{\theta} = \sum_{i=0}^{\infty}[\gamma\boldsymbol{\lambda}]^i\boldsymbol{r}$. Similar to other kernel-based policy evaluation methods [23, 24], NPDP has a computational complexity of $\mathcal{O}(n^3)$ when performed naively. However, by taking advantage of sparse matrix computations, this complexity can be reduced to $\mathcal{O}(nz)$, where $z$ is the number of non-zero elements in $(\boldsymbol{I} - \gamma\boldsymbol{\lambda})$.

## 3 Relation to Existing Methods

The second contribution of this paper is to provide a unified view of Least Squares Temporal Difference learning (LSTD), Kernelized Temporal Difference learning (KTD), Discrete-State Dynamic Programming (DSDP), and the proposed Non-Parametric Dynamic Programming (NPDP). In this section, we utilize the Galerkin methodology from Section 2.2 to re-derive the LSTD, KTD, and DSDP algorithms, and discuss how these methods compare to NPDP. A numerical comparison is given in Section 4.

## 3.1 Least Squares Temporal Difference Learning

The LSTD algorithm allows the value function $V(\boldsymbol{s})$ to be represented by a set of $m$ arbitrary basis functions $\hat{\phi}_i(\boldsymbol{s})$, see [14]. Hence, $V(\boldsymbol{s}) = \sum_{i=1}^{m} \hat{\theta}_i \hat{\phi}_i(\boldsymbol{s}) = \hat{\boldsymbol{\phi}}(\boldsymbol{s})^T \hat{\boldsymbol{\theta}}$, where $\hat{\boldsymbol{\theta}}$ is a vector of coefficients learned during policy evaluation, and $[\hat{\boldsymbol{\phi}}(\boldsymbol{s})]_i = \hat{\phi}_i(\boldsymbol{s})$. In order to re-derive the LSTD policy evaluation, the joint distribution is represented as a set of delta functions $p(\boldsymbol{s}, \boldsymbol{a}, \boldsymbol{s}') = n^{-1} \sum_{i=1}^{n} \delta_i(\boldsymbol{s}, \boldsymbol{a}, \boldsymbol{s}')$, where $\delta_i(\boldsymbol{s}, \boldsymbol{a}, \boldsymbol{s}')$ is a Dirac delta function centered on $(\boldsymbol{s}_i, \boldsymbol{a}_i, \boldsymbol{s}'_i)$. Using Galerkin's method, the integral equation is projected into the space of the basis functions $\hat{\boldsymbol{\phi}}(\boldsymbol{s})$. Thus, Eq. (3) becomes

$$\int_{\mathbb{S}} \hat{\boldsymbol{\phi}}(\boldsymbol{s}) \, p(\boldsymbol{s}) \, \hat{\boldsymbol{\phi}}(\boldsymbol{s})^T \, \hat{\boldsymbol{\theta}} \mathrm{d}\boldsymbol{s} = \int_{\mathbb{A}} \int_{\mathbb{S}} \hat{\boldsymbol{\phi}}(\boldsymbol{s}) \, p(\boldsymbol{s}, \boldsymbol{a}) \, r(\boldsymbol{s}, \boldsymbol{a}) \, \mathrm{d}\boldsymbol{s} \mathrm{d}\boldsymbol{a} + \gamma \int_{\mathbb{S}} \hat{\boldsymbol{\phi}}(\boldsymbol{s}) \, p(\boldsymbol{s}, \boldsymbol{s}') \, \hat{\boldsymbol{\phi}}(\boldsymbol{s}')^T \, \hat{\boldsymbol{\theta}} \mathrm{d}\boldsymbol{s}' \mathrm{d}\boldsymbol{s},$$

$$\sum_{i=1}^{n} \hat{\boldsymbol{\phi}}(\boldsymbol{s}_i) \, \hat{\boldsymbol{\phi}}(\boldsymbol{s}_i)^T \, \hat{\boldsymbol{\theta}} = \sum_{j=1}^{n} r(\boldsymbol{s}_j, \boldsymbol{a}_j) \, \hat{\boldsymbol{\phi}}(\boldsymbol{s}_j) + \gamma \sum_{k=1}^{n} \hat{\boldsymbol{\phi}}(\boldsymbol{s}_k) \, \hat{\boldsymbol{\phi}}(\boldsymbol{s}'_k)^T \, \hat{\boldsymbol{\theta}},$$

$$\sum_{i=1}^{n} \hat{\boldsymbol{\phi}}(\boldsymbol{s}_i) \left( \hat{\boldsymbol{\phi}}(\boldsymbol{s}_i)^T - \gamma \hat{\boldsymbol{\phi}}(\boldsymbol{s}'_i)^T \right) \hat{\boldsymbol{\theta}} = \sum_{j=1}^{n} r(\boldsymbol{s}_j, \boldsymbol{a}_j) \, \hat{\boldsymbol{\phi}}(\boldsymbol{s}_j),$$

and thus $\boldsymbol{A}\hat{\boldsymbol{\theta}} = \boldsymbol{b}$, where $\boldsymbol{A} = \sum_{i=1}^{n} \hat{\boldsymbol{\phi}}(\boldsymbol{s}_i) (\hat{\boldsymbol{\phi}}(\boldsymbol{s}_i)^T - \gamma \hat{\boldsymbol{\phi}}(\boldsymbol{s}'_i)^T)$ and $\boldsymbol{b} = \sum_{j=1}^{n} r(\boldsymbol{s}_j, \boldsymbol{a}_j) \hat{\boldsymbol{\phi}}(\boldsymbol{s}_j)$. The final weights are therefore given by

$$\hat{\boldsymbol{\theta}} = \boldsymbol{A}^{-1} \boldsymbol{b}.$$

This equation is also solved by LSTD, including the incremental updates of $\boldsymbol{A}$ and $\boldsymbol{b}$ as new samples are acquired [14]. Therefore, LSTD can be seen as computing the transitions between the basis functions using a Monte Carlo approach. However, Monte Carlo methods rely on large numbers of samples to obtain accurate results.

A key disadvantage of the LSTD method is the need to select a specific set of basis functions. The computed value function will always be a projection of the true value function into the space of these basis functions [8]. If the true value function does not lie within the space of these basis functions, the resulting approximation may be arbitrarily inaccurate, regardless of the number of acquired samples. However, using predefined basis functions only requires inverting an $m \times m$ matrix, which results in a lower computational complexity than NPDP.

The LSTD may also need to be regularized, as the inversion of $\boldsymbol{A}$ becomes ill-posed if the basis functions are too densely spaced. Regularization has a similar effect to changing the transition probabilities of the system [25].

## 3.2 Kernelized Temporal Difference Learning Methods

The proposed approach is of course not the first to use kernels for policy evaluation. Methods such as kernelized least-squares temporal difference learning [24] and Gaussian process temporal difference learning [23] have also employed kernels in policy evaluation. Taylor and Parr demonstrated that these methods differ mainly in their use of regularization [15]. The unified view of these methods is referred to as Kernelized Temporal Difference learning.

The KTD approach assumes that the reward and value functions can be represented by kernelized linear least-squares regression; i.e., $r(\boldsymbol{s}) = \boldsymbol{k}(\boldsymbol{s})^T \boldsymbol{K}^{-1} \boldsymbol{r}$ and $V(\boldsymbol{s}) = \boldsymbol{k}(\boldsymbol{s})^T \hat{\boldsymbol{\theta}}$, where $[\boldsymbol{k}(\boldsymbol{s})]_i = k(\boldsymbol{s}, \boldsymbol{s}_i)$, $[\boldsymbol{K}]_{ij} = k(\boldsymbol{s}_i, \boldsymbol{s}_j)$, $[\boldsymbol{r}]_i = r_i$, and $\hat{\boldsymbol{\theta}}$ is a weight vector. In order to derive KTD using Galerkin's method, it is necessary to again represent the joint distribution as $p(\boldsymbol{s}, \boldsymbol{a}, \boldsymbol{s}') = n^{-1} \sum_{i=1}^{n} \delta_i(\boldsymbol{s}, \boldsymbol{a}, \boldsymbol{s}')$. The Galerkin method projects the integral equation into the space of the Kronecker delta functions $[\check{\boldsymbol{\delta}}(\boldsymbol{s})]_i = \check{\delta}_i(\boldsymbol{s}, \boldsymbol{a}_i, \boldsymbol{s}'_i)$, where $\check{\delta}_i(\boldsymbol{s}, \boldsymbol{a}, \boldsymbol{s}') = 1$ if $\boldsymbol{s}' = \boldsymbol{s}'_i$, $\boldsymbol{a} = \boldsymbol{a}_i$, and $\boldsymbol{s} = \boldsymbol{s}_i$; otherwise $\check{\delta}_i(\boldsymbol{s}, \boldsymbol{a}, \boldsymbol{s}') = 0$. Thus, Eq. (3) becomes

$$\int_{\mathbb{S}} \check{\boldsymbol{\delta}}(\boldsymbol{s}) \, p(\boldsymbol{s}) \, \boldsymbol{k}(\boldsymbol{s})^T \hat{\boldsymbol{\theta}} \mathrm{d}\boldsymbol{s} = \int_{\mathbb{S}} \check{\boldsymbol{\delta}}(\boldsymbol{s}) \, p(\boldsymbol{s}) \, r(\boldsymbol{s}) \, \mathrm{d}\boldsymbol{s} + \gamma \int_{\mathbb{S}} \check{\boldsymbol{\delta}}(\boldsymbol{s}) \, p(\boldsymbol{s}, \boldsymbol{s}') \, \boldsymbol{k}(\boldsymbol{s}')^T \hat{\boldsymbol{\theta}} \mathrm{d}\boldsymbol{s}' \mathrm{d}\boldsymbol{s},$$

By substituting $p(\boldsymbol{s}, \boldsymbol{a}, \boldsymbol{s}')$ and applying the sifting property of delta functions, this equation becomes

$$\sum_{i=1}^{n} \check{\boldsymbol{\delta}}(\boldsymbol{s}_i)\boldsymbol{k}(\boldsymbol{s}_i)^T\hat{\boldsymbol{\theta}} = \sum_{j=1}^{n} \check{\boldsymbol{\delta}}(\boldsymbol{s}_j)\boldsymbol{k}(\boldsymbol{s}_j)^T\boldsymbol{K}^{-1}\boldsymbol{r} + \gamma\sum_{k=1}^{n} \check{\boldsymbol{\delta}}(\boldsymbol{s}_k)\boldsymbol{k}(\boldsymbol{s}'_k)^T\hat{\boldsymbol{\theta}},$$

and thus $\boldsymbol{K}\hat{\boldsymbol{\theta}} = \boldsymbol{r} + \gamma\boldsymbol{K}'\hat{\boldsymbol{\theta}}$, where $[\boldsymbol{K}']_{ij} = k(\boldsymbol{s}'_i, \boldsymbol{s}_j)$. The value function weights are therefore

$$\hat{\boldsymbol{\theta}} = (\boldsymbol{K} - \gamma\boldsymbol{K}')^{-1}\boldsymbol{r},$$

which is identical to the solution found by the KTD approach [15]. In this manner, the KTD approach computes a weighting $\hat{\boldsymbol{\theta}}$ such that the difference in the value at $\boldsymbol{s}_i$ and the discounted value at $\boldsymbol{s}'_i$ equals the observed empirical reward $r_i$. Thus, only the finite set of sampled states are regarded for policy evaluation. Therefore, some KTD methods, e.g. Gaussian process temporal difference learning [23], require that the samples are obtained from a single trajectory to ensure that $\boldsymbol{s}'_i = \boldsymbol{s}_{i+1}$.

A key difference between KTD and NPDP is the representation of the value function $V(\boldsymbol{s})$. The form of the value function is a direct result of the representation used to embody the state transitions. In the original paper [15], the KTD algorithm represents the transitions by using linear kernelized regression $\hat{\boldsymbol{k}}(\boldsymbol{s}') = \boldsymbol{k}(\boldsymbol{s})^T\boldsymbol{K}^{-1}\boldsymbol{K}'$, where $[\hat{\boldsymbol{k}}(\boldsymbol{s}')]_i = \mathbb{E}[k(\boldsymbol{s}', \boldsymbol{s}_i)]$. The value function $V(\boldsymbol{s}) = \boldsymbol{k}(\boldsymbol{s})^T\hat{\boldsymbol{\theta}}$ is the correct form for this transition model. However, the transition model does not explicitly represent a conditional distribution and can lead to inaccurate predictions. For example, consider two samples that start at $s_1 = 0$ and $s_2 = 0.75$ respectively, and both transition to $s' = 0.75$. For clarity, we use a box-cart kernel with a width of one $k(s_i, s_j) = 1$ iff $\|s_i - s_j\| \leq 0.5$ and 0 otherwise. Hence, $\boldsymbol{K} = \boldsymbol{I}$ and each row of $\boldsymbol{K}$' corresponds to $(0, 1)$. In the region $0.25 \leq s \leq 0.5$, where the two kernels overlap, the transition model would then predict $\hat{k}(s) = \boldsymbol{k}(\boldsymbol{s})^T\boldsymbol{K}^{-1}\boldsymbol{K}' = [\ 0\ \ 2\ ]$. This prediction is however impossible as it requires that $\mathbb{E}[k(s', s_2)] > \max_s k(s, s_2)$. In comparison, NPDP would predict the distribution $\psi(s') \equiv \psi_1(s') \equiv \psi_2(s')$ for all states in the range $-0.5 \leq s \leq 1.25$.

Similar as for LSTD, the matrix $(\boldsymbol{K} - \gamma\boldsymbol{K}')$ may become singular and thus not be invertible. As a result, KTD usually needs to be regularized [15]. Given that KTD requires inverting an $n \times n$ matrix, this approach has a computational complexity similar to NPDP.

### 3.3 Discrete-State Dynamic Programming

The standard tabular DSDP approach can also be derived using the Galerkin method. Given a system with $q$ discrete states, the value function has the form $V(\boldsymbol{s}) = \check{\boldsymbol{\delta}}(\boldsymbol{s})^T\boldsymbol{v}$, where $\check{\boldsymbol{\delta}}(\boldsymbol{s})$ is a vector of $q$ Kronecker delta functions centered on the discrete states. The corresponding reward function is $r(\boldsymbol{s}) = \check{\boldsymbol{\delta}}(\boldsymbol{s})^T\bar{\boldsymbol{r}}$. The joint distribution is given by $p(\boldsymbol{s}', \boldsymbol{s}) = q^{-1}\boldsymbol{\delta}(\boldsymbol{s})^T\boldsymbol{P}\boldsymbol{\delta}(\boldsymbol{s}')$, where $\boldsymbol{P}$ is a stochastic matrix $\sum_{j=1}^{q}[\boldsymbol{P}]_{ij} = 1$, $\forall i$ and hence $p(\boldsymbol{s}) = q^{-1}\sum_{i=1}^{q}\delta_i(\boldsymbol{s})$. Galerkin's method projects the integral equation into the space of the states $\check{\boldsymbol{\delta}}(\boldsymbol{s})$. Thus, Eq. (3) becomes

$$\int_{\mathbb{S}} \check{\boldsymbol{\delta}}(\boldsymbol{s})\,p(\boldsymbol{s})\,\check{\boldsymbol{\delta}}(\boldsymbol{s})^T\boldsymbol{v}\mathrm{d}\boldsymbol{s} = \int_{\mathbb{S}} \check{\boldsymbol{\delta}}(\boldsymbol{s})\,p(\boldsymbol{s})\,\check{\boldsymbol{\delta}}(\boldsymbol{s})^T\bar{\boldsymbol{r}}\mathrm{d}\boldsymbol{s} + \gamma\int_{\mathbb{S}} \check{\boldsymbol{\delta}}(\boldsymbol{s})\,p(\boldsymbol{s}, \boldsymbol{s}')\,\check{\boldsymbol{\delta}}(\boldsymbol{s}')^T\boldsymbol{v}\mathrm{d}\boldsymbol{s}'\mathrm{d}\boldsymbol{s},$$

$$\boldsymbol{I}\boldsymbol{v} = \boldsymbol{I}\bar{\boldsymbol{r}} + \gamma\int_{\mathbb{S}} \check{\boldsymbol{\delta}}(\boldsymbol{s})\,\boldsymbol{\delta}(\boldsymbol{s})^T\boldsymbol{P}\boldsymbol{\delta}(\boldsymbol{s}')\check{\boldsymbol{\delta}}(\boldsymbol{s}')^T\boldsymbol{v}\mathrm{d}\boldsymbol{s}'\mathrm{d}\boldsymbol{s},$$

$$\boldsymbol{v} = \bar{\boldsymbol{r}} + \gamma\boldsymbol{P}\boldsymbol{v},$$

$$\boldsymbol{v} = (\boldsymbol{I} - \gamma\boldsymbol{P})^{-1}\bar{\boldsymbol{r}}, \tag{6}$$

which is the same computation used by DSDP [16]. The DSDP and NPDP methods actually use similar models to represent the system. While NPDP uses a kernel density estimation, the DSDP algorithm uses a histogram representation. Hence, DSDP can be regarded as a special case of NPDP for discrete state systems.

The DSDP algorithm has also been the basis for continuous-state policy evaluation algorithms [26, 27]. These algorithms first use the sampled states as the discrete states of an MDP and compute the corresponding values. The computed values are then generalized, under a smoothness assumption, to the rest of the state-space using local averaging. Unlike these methods, NPDP explicitly performs policy evaluation for a continuous set of states.

# 4  Numerical Evaluation

In this section, we compare the different policy evaluation methods discussed in the previous section, with the proposed NPDP method, on an illustrative benchmark system.

## 4.1  Benchmark Problem and Setup

In order to compare the LSTD, KTD, DSDP, and NPDP approaches, we evaluated the methods on a discrete-time continuous-state system. A standard linear-Gaussian system was used for the benchmark problem, with transitions given by $s' = 0.95s + \omega$ where $\omega$ is Gaussian noise $\mathcal{N}(\mu = 0, \sigma = 0.025)$. The initial states are restricted to the range 0.95 to 1. The reward functions consist of three Gaussians, as shown by the black line in Fig. 1.

The KTD method was implemented using a Gaussian kernel function and regularization. The LSTD algorithm was implemented using 15 uniformly-spaced normalized Gaussian basis functions, and did not require regularization. The DSDP method was implemented by discretizing the state-space into 10 equally wide regions. The NPDP method was also implemented using Gaussian kernels.

The hyper-parameters of all four methods, including the number of basis functions for LSTD and DSDP, were carefully tuned to achieve the best performance. As a performance base-line, the values of the system in the range $0 < s < 1$ were computed using a Monte Carlo estimate based on 50000 trajectories. The policy evaluations performed by the tested methods were always based on only 500 samples; i.e. 100 times less samples than the base-line. The experiment was run 500 times using independent sets of 500 samples. The samples were not drawn from the same trajectory.

## 4.2  Results

The performance of the different methods were compared using three performance measures. Two of the performance measures are based on the weighted *Mean Squared Error* (MSE) [2] $E(V) = \int_0^1 W(s)\left(V(s) - V^\star(s)\right)^2 \mathrm{d}s$ where $V^\star$ is the true value function and $W(s) \geq 0$, for all states, is a weighting distribution $\int_0^1 W(s)\mathrm{d}s = 1$. The first performance measure $E_{\mathrm{unif}}$ corresponds to the MSE where $W(s) = 1$ for all states in the range zero to one. The second performance measure $E_{\mathrm{samp}}$ corresponds to the MSE where $W(s) = n^{-1}\Sigma_{i=1}^n \delta_i(s)$ respectively. Thus, $E_{\mathrm{samp}}$ is an indicator of the accuracy in the space of the samples, while $E_{\mathrm{unif}}$ is an indicator of how well the computed value function generalizes to the entire state space. The third performance measure $E_{\mathrm{max}}$ is given by the maximum error in the value function. This performance measure is the basis of a bound on the overall value function approximation [20].

The results of the experiment are shown in Table 1. The performance measures were averaged over the 500 independent trials of the experiment. For all three performance measures, the NPDP algorithm achieved the highest levels of performance, while the DSDP approach consistently led to the worst performance.

|  | $\mathbf{E}_{\mathrm{unif}}$ | $\mathbf{E}_{\mathrm{samp}}$ | $\mathbf{E}_{\mathrm{max}}$ |
|---|---|---|---|
| NPDP | $\mathbf{0.5811} \pm 0.0333$ | $\mathbf{0.7185} \pm 0.0321$ | $\mathbf{1.4971} \pm 0.0309$ |
| LSTD | $0.6898 \pm 0.0443$ | $0.8932 \pm 0.0412$ | $1.5591 \pm 0.0382$ |
| KTD | $0.7585 \pm 0.0460$ | $0.8681 \pm 0.0270$ | $2.5329 \pm 0.0391$ |
| DSDP | $1.6979 \pm 0.0332$ | $2.1548 \pm 0.1082$ | $2.9985 \pm 0.0449$ |

Table 1: Each row corresponds to one of the four tested algorithms for policy evaluation. The columns indicate the performance of the approaches during the experiment. The performance indexes include the mean squared error evaluated uniformly over the zero to one range, the mean squared error evaluated at the 500 sampled points, and the maximum error. The results are averaged over 500 trials. The standard errors of the means are also given.

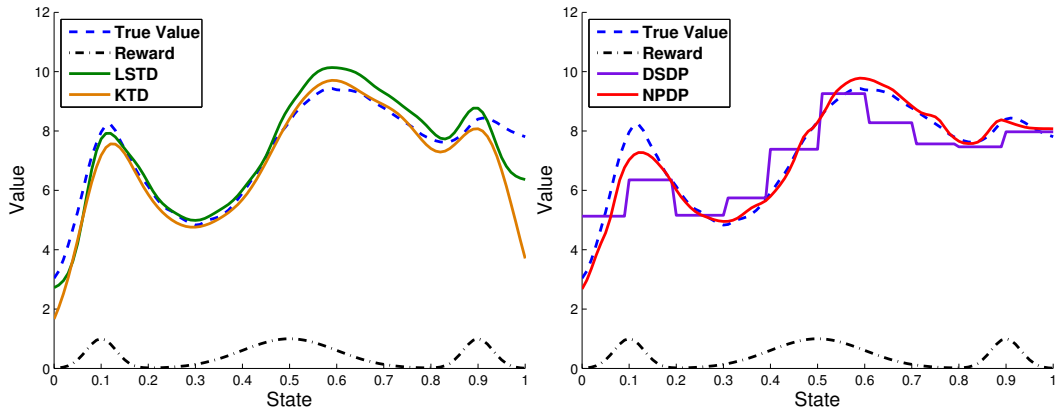

Figure 1: Value functions obtained by the evaluated methods. The black lines show the reward function. The blue lines show the value function computed from the trajectories of 50,000 uniformly sampled points. The LSTD, KTD, DSDP, and NPDP methods evaluated the policy using only 500 points. The presentation was divided into two plots for improved clarity

## 4.3   Discussion

The LSTD algorithm achieved a relatively low $E_{\text{unif}}$ value, which indicates that the tuned basis functions could accurately represent the true value function. However, the performance of LSTD is sensitive to the choice of basis functions and the number of samples per basis function. Using 20 basis functions instead of 15 reduces the performance of LSTD to $E_{\text{unif}} = 2.8705$ and $E_{\text{samp}} = 1.0256$ as a result of overfitting. The KTD method achieved the second best performance for $E_{\text{samp}}$, as a result of using a non-parametric representation. However, the value tended to drop in sparsely-sampled regions, which lead to relatively high $E_{\text{unif}}$ and $E_{\text{max}}$ values. The discretization of states for DSDP is generally a disadvantage when modeling continuous systems, and resulted in poor overall performance for this evaluation. The NPDP approach out-performed the other methods in all three performance measures. The performance of NPDP could be further improved by using adaptive kernel density estimation [28] to locally adapt the kernels' bandwidths according to the sampling density. However, all methods were restricted to using a single global bandwidth for the purpose of this comparison.

## 5   Conclusion

This paper presents two key contributions to continuous-state policy evaluation. The first contribution is the Non-Parametric Dynamic Programming algorithm for policy evaluation. The proposed method uses a kernel density estimate to generate a consistent representation of the system. It was shown that the true form of the value function for this model is given by a Nadaraya-Watson kernel regression. The NPDP algorithm provides a solution for calculating the value function. As a kernel-based approach, NPDP simultaneously addresses the problems of function approximation and policy evaluation.

The second contribution of this paper is providing a unified view of Least-Squares Temporal Difference learning, Kernelized Temporal Difference learning, and discrete-state Dynamic Programming, as well as NPDP. All four approaches can be derived from the Bellman equation using the Galerkin projection method. These four approaches were also evaluated and compared on an empirical problem with a continuous state space and non-linear reward function, wherein the NPDP algorithm out-performed the other methods.

## Acknowledgements

The project receives funding from the European Community's Seventh Framework Programme under grant agreement n° ICT- 248273 GeRT and n° 270327 Complacs.

# References

[1] Dimitri P. Bertsekas. *Dynamic Programming and Optimal Control, Vol. II.* Athena Scientific, 2007.

[2] R. S. Sutton and A. G. Barto. *Reinforcement Learning: An Introduction.* 1998.

[3] H. Maei, C. Szepesvari, S. Bhatnagar, D. Precup, D. Silver, and R. Sutton. Convergent temporal-difference learning with arbitrary smooth function approximation. In *NIPS*, pages 1204–1212, 2009.

[4] Richard Bellman. Bottleneck problems and dynamic programming. *Proceedings of the National Academy of Sciences of the United States of America*, 39(9):947–951, 1953.

[5] R.E. Kalman. Contributions to the theory of optimal control, 1960.

[6] Warren B. Powell. *Approximate Dynamic Programming: Solving the Curses of Dimensionality (Wiley Series in Probability and Statistics).* Wiley-Interscience, 2007.

[7] Rémi Munos. Geometric Variance Reduction in Markov Chains: Application to Value Function and Gradient Estimation. *Journal of Machine Learning Research*, 7:413–427, 2006.

[8] Ralf Schoknecht. Optimality of reinforcement learning algorithms with linear function approximation. In *NIPS*, pages 1555–1562, 2002.

[9] Leemon Baird. Residual algorithms: Reinforcement learning with function approximation. In *ICML*, 1995.

[10] Christopher G. Atkeson and Juan C. Santamaria. A Comparison of Direct and Model-Based Reinforcement Learning. In *ICRA*, pages 3557–3564, 1997.

[11] H. Bersini and V. Gorrini. Three connectionist implementations of dynamic programming for optimal control: A preliminary comparative analysis. In *Nicrosp*, 1996.

[12] E. Nadaraya. On estimating regression. *Theory of Prob. and Appl.*, 9:141–142, 1964.

[13] G. Watson. Smooth regression analysis. *Sankhya, Series*, A(26):359–372, 1964.

[14] Justin A. Boyan. Least-squares temporal difference learning. In *ICML*, pages 49–56, San Francisco, CA, USA, 1999. Morgan Kaufmann Publishers Inc.

[15] Taylor, Gavin and Parr, Ronald. Kernelized value function approximation for reinforcement learning. In *ICML*, pages 1017–1024, New York, NY, USA, 2009. ACM.

[16] Dimitri P. Bertsekas and John N. Tsitsiklis. *Neuro-Dynamic Programming.* Athena Scientific, 1996.

[17] Murray Rosenblatt. Remarks on Some Nonparametric Estimates of a Density Function. *The Annals of Mathematical Statistics*, 27(3):832–837, September 1956.

[18] Emanuel Parzen. On Estimation of a Probability Density Function and Mode. *The Annals of Mathematical Statistics*, 33(3):1065–1076, 1962.

[19] G. S. Kimeldorf and G. Wahba. Some results on Tchebycheffian spline functions. *Journal of Mathematical Analysis and Applications*, 33(1):82–95, 1971.

[20] Rémi Munos. Error bounds for approximate policy iteration. In *ICML*, pages 560–567, 2003.

[21] Kendall E. Atkinson. *The Numerical Solution of Integral Equations of the Second Kind.* Cambridge University Press, 1997.

[22] Dominik Wied and Rafael Weissbach. Consistency of the kernel density estimator: a survey. *Statistical Papers*, pages 1–21, 2010.

[23] Yaakov Engel, Shie Mannor, and Ron Meir. Reinforcement learning with Gaussian processes. In *ICML*, pages 201–208, New York, NY, USA, 2005. ACM.

[24] Xin Xu, Tau Xie, Dewen Hu, and Xicheng Lu. Kernel least-squares temporal difference learning. *International Journal of Information Technology*, 11:54–63, 1997.

[25] J. Zico Kolter and Andrew Y. Ng. Regularization and feature selection in least-squares temporal difference learning. In *ICML*, pages 521–528. ACM, 2009.

[26] Nicholas K. Jong and Peter Stone. Model-based function approximation for reinforcement learning. In *AAMAS*, May 2007.

[27] Dirk Ormoneit and Śaunak Sen. Kernel-Based reinforcement learning. *Machine Learning*, 49(2):161–178, November 2002.

[28] B. W. Silverman. *Density estimation: for statistics and data analysis.* London, 1986.

